# Attractor Neural Networks with Local Inhibition: from Statistical Physics to a Digital Programmable Integrated Circuit

**E. Pasero**
Dipartimento di Elettronica
Politecnico di Torino
I-10129 Torino, Italy

**R. Zecchina**
Dipartimento di Fisica Teorica e INFN
Universitá di Torino
I-10125 Torino, Italy

## Abstract

Networks with local inhibition are shown to have enhanced computational performance with respect to the classical Hopfield-like networks. In particular the critical capacity of the network is increased as well as its capability to store correlated patterns. Chaotic dynamic behaviour (exponentially long transients) of the devices indicates the overloading of the associative memory. An implementation based on a programmable logic device is here presented. A 16 neurons circuit is implemented whit a XILINK 4020 device. The peculiarity of this solution is the possibility to change parts of the project (weights, transfer function or the whole architecture) with a simple software download of the configuration into the XILINK chip.

## 1  INTRODUCTION

Attractor Neural Networks endowed with local inhibitory feedbacks, have been shown to have interesting computational performances[1]. Past effort was concentrated in studying a variety of synaptic structures or learning algorithms, while less attention was devoted to study the possible role played by different dynamical schemes. The definition of relaxation dynamics is the central problem for the study of the associative and computational capabilities in models of attractor neural networks and might be of interest also for hardware implementation in view of the

constraints on the precision of the synaptic weights.

In this paper, we give a brief discussion concerning the computational and physical role played by local inhibitory interactions which lead to an effective non-monotonic transfer function for the neurons. In the last few years others models characterized by non-monotonic neurons have been proposed[2,3].

For Hebbian learning we show, numerically, that the critical capacity increases with respect to the Hopfield case and that such result can be interpreted in terms of a twofold task realized by the dynamical process. By means of local inhibition, the system dynamically selects a subspace (or subnetwork) of minimal static noise with respect to the recalled pattern; at the same time, and in the selected subspace, the retrieval of the memorized pattern is performed. The dynamic behaviour of the network, for deterministic sequential updating, range from fixed points to chaotic evolution, with the storage ratio as control parameter, the transition appearing in correspondence to the collapse of the associative performance. Resorting to two simplified versions of the model, we study the problem of their optimal performance by the replica method; in particular the role of non-monotonic functions and of subspaces dynamical selection are discussed.

In a second part of the work, the implementation of the discussed model by means of a XILINK programmable gate array is discussed. The circuit implements a 16-32 neurons network in which the analogical characteristics (such as a capacitive decay) are emulated by digital solutions. As expected, the limited resolution of the weghts does not represent a limit for the performance of the network.

## 2   THE MODEL: theory and performance

We study an attractor neural network composed of $N$ three state $\pm 1, 0$ formal neurons. The $\pm 1$ values code for the patterns (the patterns are indeed binary) and are thus used during the learning phase, while the 0-state is a *don't care* state, not belonging to the patterns code, which has only a dynamical role. The system is assumed to be fully connected and its evolution is governed by sequential or parallel updating of the following equations

$$S_i = \begin{cases} sgn(h_i) & if \quad |h_i| \leq \gamma \\ 0 & if \quad |h_i| > \gamma \end{cases} \tag{1}$$

$$h_i(t+1) = \lambda h_i(t) + \sum_{j=1}^{N} J_{ij} S_j(t) \qquad i = 1, ..., N \tag{2}$$

where $\gamma$ is a dynamic threshold of the local inhibitory feedback (typically we take $\gamma(t) = \frac{1}{N} \sum_i |h_i(t-1)|$), the $\{J_{ij}\}$ are the *synaptic* conductances and $\lambda$ is a capacitive decay factor of the input potential ($\lambda = e^{\frac{1}{\tau}}$, where $\tau = RC$).

The performance of the network are described in terms of two parameters which have both a dynamical and a computational simple interpretation. In particular we define the **retrieval activity** as the fraction of neurons which are not not in the zero state

$$a = \frac{1}{N} \sum_i S_i^2, \tag{3}$$

while the parameter that defines the retrieval quality is the **scaled overlap**

$$m^\mu = \frac{1}{Na} \sum_i \xi_i^\mu S_i. \tag{4}$$

where the $\{\xi_i^\mu = \pm 1, \ i = 1, N; \mu = 1, P\}$ are the memorized binary patterns. The scaled overlap can be thought simply as the overlap computed in the subspace $M$ of the *active* neurons, $M \equiv \{i \ / \ S_i \neq 0, \ i = 1, N\}$.

Given a set of $P$ random independent binary patterns $\{\xi_i^\mu\}$, the Hebb-Hopfield learning rule corresponds to fix the synaptic matrix $J_{ij}$ by the additive relation $J_{ij} = \frac{1}{N} \sum_{\mu=1}^{P} \xi_i^\mu \xi_j^\mu$ (with $J_{ii} = 0$). The effect of the dynamical process defined by (1) and (2) is the selection of subspaces $M$ of active neurons in which the static noise is minimized (such subspaces will be hereafter referred to as *orthogonal* subspaces). Before entering in the description of the results, it is worthwhile to remember that, in Hopfield-like attractor neural networks, the mean of cross correlation fluctuations produce in the local fields of the neurons a static noise, referred to as cross-talk of the memories. Together with temporal correlations, the static noise is responsible of the phase transition of the neural networks from associative memory to spin-glass. More precisely, when the Hopfield model is in a fixed point $\xi^\sigma$ which belongs to the set of memories, the local fields are given by $h_i \xi_i^\sigma = 1 + R_i^\sigma$ where $R_i^\sigma = \frac{1}{N} \sum_{\mu \neq \sigma} \sum_{j \neq i} \xi_i^\mu \xi_j^\mu \xi_i^\sigma \xi_j^\sigma$ is the static noise (gaussian distribution with 0 mean and variance $\sqrt{\alpha}$).

The preliminary performance study of the model under discussion have revealed several new basic features, in particular: (i) the critical capacity, for the Hebb learning rule, results increased up to $\alpha_c \approx 0.33$ (instead of 0.14[4]); (ii) the mean cross correlation fluctuations computed in the selected subspaces is minimized by the dynamical process in the region $\alpha < \alpha_c$; (iii) in correspondence to the associative transition the system goes through a dynamic transition from fixed points to chaotic trajectories.

The quantitative results concerning associative performance, are obtained by means of extended simulations. A typical simulation takes the memorized patterns as initial configurations and lets the system relax until it reaches a stationary point. The quantity describing the performance of the network as an associative memory is the mean scaled overlap $m$ between the final stationary states and the memorized patterns, used as initial states. As the number of memorized configurations grows, one observes a threshold at $\alpha = \alpha_c \approx 0.33$ beyond which the stored states become unstable. (numerical results were performed for networks of size up to $N = 1000$). We observe that since the recall of the patterns is performed with no errors (up to

$\alpha \approx 0.31$), also the number of stored bits in the synaptic matrix results increased with respect to the Hopfield case.

The typical size of the sub-networks $D_M$, like the network capacity, depends on the threshold parameter $\gamma$ and on the kind of updating: for $\gamma(t) = \frac{1}{N} \sum_i |h_i(t-1)|$ and parallel updating we find $D_M \simeq N/2$ ($\alpha_c = 0.33$).

The static noise reduction corresponds to the minimization of the mean fluctuation of the cross correlations (cross talk) in the subspaces, defined by

$$C = \frac{1}{P} \sum_\sigma \sum_{\mu \neq \sigma} (\frac{1}{aN} \sum_{i=1}^{N} \xi_i^\mu \xi_i^\sigma \epsilon_i^\sigma)^2 = \frac{1}{P} \sum_\sigma \sum_{\mu \neq \sigma} (\frac{1}{aN} \sum_{i \in M} \xi_i^\mu \xi_i^\sigma)^2 \qquad (5)$$

where $\epsilon_i^\sigma = 1$ if $i \in M$ in pattern $\sigma$ and zero otherwise, as a function of $\alpha$. Under the dynamical process (1) and (2), $C$ does not follow a statistical law but undergoes a minimization that qualitatively explains the increase in the storage capacity. For $\alpha < \alpha_c$, once the system has relaxed in a stationary subspace, the model becomes equivalent (in the subspace) to a Hopfield network with a static noise term which is no longer random. The statistical mechanics of the combinatorial task of minimizing the *noise-energy* term (5) can be studied analytically by the replica method; the results are of general interest in that give an upper bound to the performance of networks endowed with Hebb-like synaptic matrices and with the possibility selecting optimal subnetworks for retrieval dynamics of the patterns[8].

As already stated, the behaviour of the neural network as a dynamical system is directly related to its performance as an associative memory. The system shows an abrupt transition in the dynamics, from fixed points to chaotic exponentially long transients, in correspondence to the value of the storage ratio at which the memorized configurations become unstable. The only (external) control parameter of the model as a dynamical system is the storage ratio $\alpha = P/N$. Dynamic complex behaviour appears as a clear signal of saturation of the attractor neural network and does not depend on the symmetry of the couplings.

As a concluding remark concerning this short description of the network performance, we observe that the dynamic selection of subspaces seems to take advantage of finite size effects allowing the storage of correlated patterns also with the simple Hebb rule. Analytical and numerical work is in progress on this point, devoted to clarify the performance with spatially correlated patterns[5].

Finally, we end this theoretical section by addressing the problem of optimal performance for a different choice of the synaptic weights. In this direction, it is of basic interest to understand whether a dynamical scheme which allows for dynamic selection of subnetworks provides a neural network model with enhanced optimal capacity with respect to the classical spin models. Assuming that nothing is known about the couplings, one can consider the $J_{ij}$ as dynamical variables and study the fractional volume in the space of interactions that makes the patterns fixed points of the dynamics. Following Gardner and Derrida[6], we describe the problem in terms of a cost-energy function and study its statistical mechanics: for a generic choice of the $\{J_{ij}\}$, the cost function $E_i$ is defined to be the number of patterns such that a given site $i$ is wrong (with respect to (1))

$$E_i(\{J_{ij}\}, \{\epsilon_i^\mu\}) = \sum_{\mu=1}^{P} \left[ \epsilon_i^\mu \left( \Theta(h_i^\mu \xi_i^\mu + \gamma) - \Theta(h_i^\mu \xi_i^\mu) \right) + (1 - \epsilon_i^\mu)\Theta(\gamma^2 - h_i^{\mu 2}) \right] \quad (6)$$

where $\Theta$ is the step function, the $h_i^\mu = \dfrac{1}{\sqrt{N}} \sum_j J_{ij} \xi_j^\mu \epsilon_j^\mu$ are the local fields, $\gamma$ is the threshold of the inhibitory feedback and with $\epsilon_i^\mu = \{0, 1\}$ being the variables that identify the subspace $M$ ($\epsilon_i^\mu = 1$ if $i \in M$ and zero otherwise).

In order to estimate the optimal capacity, one should perform the replica theory on the following partition function

$$Z = \text{Tr}_{\{\epsilon_i^\mu / \sum_i \epsilon_i^\mu = aN\}} \int \prod_{i \neq j} dJ_{ij} \delta(\sum_{j(\neq i)} J_{ij}^2 - N) e^{-\beta \sum_i E_i} \quad (7)$$

Since the latter task seems unmanageable, as a first step we resort to two simplified version of the model which, separately, retain its main characteristics (subspaces and non-monotonicity); in particular:

(i) we assume that the $\{\epsilon_i^\mu\}$ are quenched random variables, distributed according to $P(\epsilon_i^\mu) = (1 - A)\delta(\epsilon_i^\mu) + A\delta(\epsilon_i^\mu - 1)$, $A \in [0, 1]$;

(ii) we consider the case of a two-state ($\pm 1$) non-monotonic transfer function.

For lack of space, here we list only the final results. The expressions of the R.S. critical capacity for the models are, respectively:

$$\alpha_c^{R.S.}(\gamma; A) = \left\{ 2(1 - A) \int_0^\gamma D\zeta(\gamma - \zeta)^2 + \frac{A}{2} + A \int_\gamma^\infty D\zeta(\gamma - \zeta)^2 \right\}^{-1} \quad (8)$$

$$\alpha_c^{R.S.}(\gamma) = \left\{ \int_0^{\frac{\gamma}{2}} D\zeta \zeta^2 + \int_{\frac{\gamma}{2}}^\infty D\zeta(\gamma - \zeta)^2 \right\}^{-1} \quad (9)$$

where $D\zeta = \dfrac{1}{\sqrt{2\pi}} e^{\frac{-\zeta^2}{2}} d\zeta$ (for (9) see also Ref.[4]).

The values of critical capacity one finds are much higher than the monotonic perceptron capacity ($\alpha_c = 2$). Unfortunately, the latter results are not reliable in that the stability analysis shows that the RS solution are unstable. Replica symmetry breaking is thus required. All the details concerning the computation with one step in replica symmetry breaking of the critical capacity and stabilities distribution can be found in Ref.[9]. Here we just quote the final quantitative result concerning optimal capacity for the non-monotonic two-state model: numerical evaluation of the saddle-point equations (for unbiased patterns) gives $\alpha_c(\gamma^{opt}) \approx 4.8$ with $\gamma^{opt} \approx 0.8$, the corresponding R.S. value from (9) being $\alpha_c^{R.S.} \approx 10.5$.

## 3    HARDWARE IMPLEMENTATION: a digital programmable integrated circuit

The performance of the network discussed in the above section points out the good behavior of the *dynamical* approach. Our goal is now to investigate the performance of this system with special hardware. Commercial neural chips[9] and[10] are not feasible: the featu res of our net require non monotonic transfer characteristic due to local inhibitions. These aspects are not allowed in traditional networks. The implementation of a full custom chip is, on the other side, an hasty choice. The model is still being studied: new developments must be expected in the next future. Therefore we decided to build a prototype based on programmable logic circuits. This solution allows us to implement the circuit in a short time not to the detriment of the performance. Moreover the same circuit will be easily updated to the next evolutions. After an analysis of the existing logic circuits we decided to use the FPGAs devices[11]. The reasons are that we need a large quantity of internal registers, to represent both synapses and capacitors, and the fastest interconnections. Xilinx 4000 family[12] offers us the most interesting approach: up tp 20000 gates are programmable and up to 28 K bit of Ram are available. Moreover a 3 ns propagation delay between internal blocks allow to implement very fast systems. We decided to use a XC4020 circuit with 20000 equivalent gates. Main problems related to the implementation of our model are the following: (a) number of neurons, (b) number of connections and (c) computation time parameters. (a) and (b) are obviously related to the logic device we have at our disposal. The number of gates we can use to implement the transfer function of our non monotonic neurons are mutually exclusive with the number of bits we decide to assign to the weights. The 20000 gates must be divided between logic gates and Ram cells. The parameter (c) depends on our choices in implementing the neural network. We can decide to connect the logic blocks in a sequential or in a parallel way. The global propagation time is the sum of the propagation delays of each logic block, from the the input to the output. Therefore if we put more blocks in parallel we don't increase the propagation delay and the time performance is better. Unfortunately the parallel solution clashes with the limitations of available logic blocks of our device. Therefore we decided to design two chips: the first circuit can implement 16 neurons in a faster parallell implementation and the second circuit allow us to use 32 (or more) neurons in a slower serial approach. Here we'll describe the fastest implementation.

Figure 1 shows the 16 neurons of the neural chip. Each neuron, described in figure 2, performs a sequential sum and multiplication of the outputs of the other 15 neurons by the synaptic values stored inside the internal Ram. A special circuit implements the activation function described in the previuos section. All the neurons perform these operations in a parallel way: 15 clock pulses are sufficient to perform the complete operation for the system. Figure 2 shows the circuit of the neuron. T1 is a Ram where the synapses Tij are stored after the training phase. M1 and A1 perform sums and multiplications according to our model. D1 simulates the $\lambda$ decay factor: every 15 clock cycles, which correspond to a complete cycle of sum and multiplication for all the 16 neurons, this circuit decreases the input of the neuron of a factor $\lambda$. The activation function (1) is realized by F1. Such circuit emulates a three levels logic, based on -1, 0 and +1 values by using two full adder blocks. Limitation due to the electrical characteristic of the circuit, impose a maximum

clock cycle of 20 MHz . The 16 neurons version of the chip takes from 4 to 6 complete computations to gain stability and every computation is 16 clock cycles long. Therefor e the network gives a stable state after 3 $\mu s$ at maximum. The second version of this circuit allows to use more neurons at a lower speed. We used the Xilinx device to implement one neuron while the synapses and the capacitors are stored in an external fast memory. The single neuron is time multiplexed in order to emulate a large number identical devices. At each step, both synapses and state variables are downloaded and uploaded from an external memory. This solution is obviously slower than the first one but a larger number of nerons can be implemented. A 32 neurons version takes about 6 $\mu s$ to reach a stable configuration.

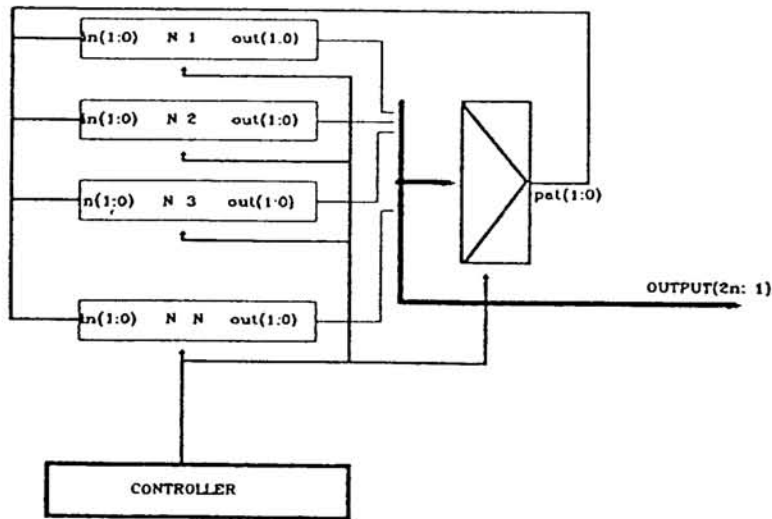

Figure 1  -  Neural Chip

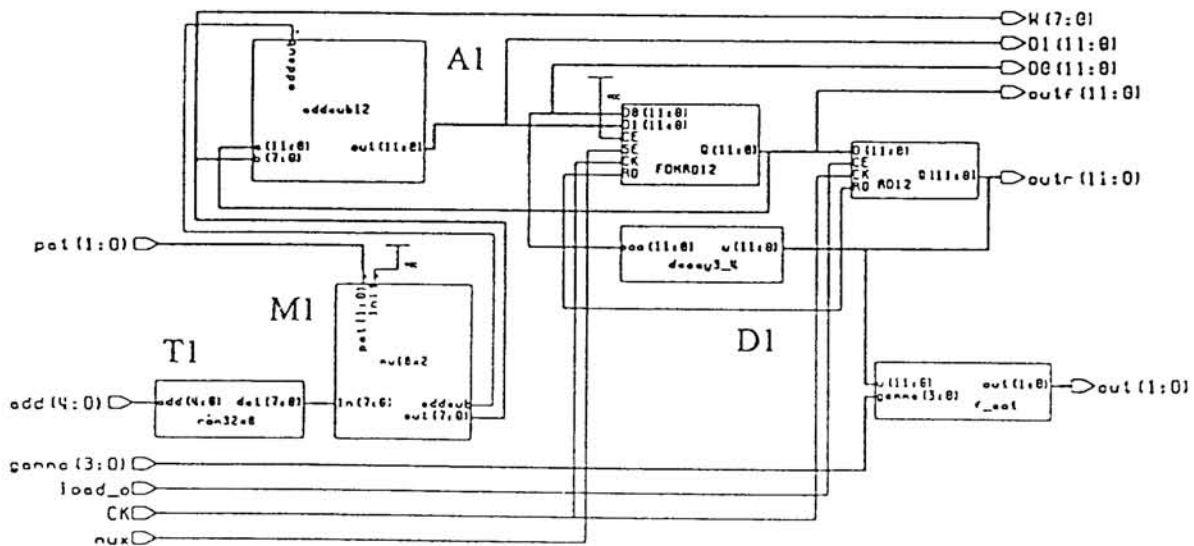

Figure 2  -  Neuron

## 4   CONCLUSION

A modified approach to attractor neural networks and its implementation on a XILINK XC4020 FPGA was discussed. The chip is now under test. Six $\mu s$ are sufficient for the relaxation of the system in a stable state, and the recognition of an input pattern is thus quite fast. A next step will be the definition of a multiple chip system endowed with more than 32 neurons, with the weights stored in an external fast memory.

### Acknowledgements

This work was partially supported by the *Annethe*-INFN Italian project and by *Progetto finalizzato sistemi informatici e calcolo parallelo* of CNR under grant N. 91.00884.PF69.

# References

[1] R. Zecchina, "Computational and Physical Role of Local Inhibition in Attractor Neural Networks: a Simple Model," *Parallel Architectures and Neural Networks*, edt. E.R. Caianiello, World Scientific (1992).

[2] M. Morita, S. Yoshizawa, H. Nakano, "Analysis and Improvement of the Dynamics of Autocorrelated Associative Memory," *IEICE Trans.* **J73-D-II**, 232 (1990).

[3] K. Kobayashi, "On the Capacity of a Neuron with a Non-Monotone Output Function," *Network*, **2**, 237 (1991).

[4] D.J. Amit, M. Gutfreund, H. Sompolinsky, "Storing Infinite Numbers of Patterns in a Spin-Glass Model of Neural Networks," *Phys. Rev. Lett.*, **55**, 1530 (1985).

[5] G. Boffetta, N. Brunel, R. Monasson, R. Zecchina, in preparation (1993).

[6] E. Gardner, B. Deridda, "Optimal Storage Properties of Neural Network Models," *J. Phys.*, **A21**, 271 (1988).

[7] G. Boffetta, R. Monasson, R. Zecchina, "Symmetry Breaking in Non-Monotonic Neural Networks," in preparation (1992).

[8] N. Brunel, R. Zecchina, "Statistical Mechanics of Optimal Memory Retrieval in the Space of Dynamic Neuronal Activities," preprint (1993).

[9] "An electrical Trainable Artificial Neural Network", proceedings of IJCNN, 1989, S. Diego.

[10] M. Dzwonczyk, M.Leblanc, "INCA: An Integrated Neurocomputing Architecture", proceedings of AIAA Computing in Aerospace, October 1991

[11] W.R. Moore, W. Luk, "FPGAs", Abingdon EE-CS Books, 1991

[12] "The XC4000 Data Book", Xilinx, 1991
